# Statistical Modeling of Cell-Assemblies Activities in Associative Cortex of Behaving Monkeys

**Itay Gat and Naftali Tishby**
Institute of Computer Science and
Center for Neural Computation
Hebrew University, Jerusalem 91904, Israel *

## Abstract

So far there has been no general method for relating extracellular electrophysiological measured activity of neurons in the associative cortex to underlying network or "cognitive" states. We propose to model such data using a multivariate Poisson Hidden Markov Model. We demonstrate the application of this approach for temporal segmentation of the firing patterns, and for characterization of the cortical responses to external stimuli. Using such a statistical model we can significantly discriminate two behavioral modes of the monkey, and characterize them by the different firing patterns, as well as by the level of coherency of their multi-unit firing activity.

Our study utilized measurements carried out on behaving Rhesus monkeys by M. Abeles, E. Vaadia, and H. Bergman, of the Hadassa Medical School of the Hebrew University.

## 1 Introduction

Hebb hypothesized in 1949 that the basic information processing unit in the cortex is a cell-assembly which may include thousands of cells in a highly interconnected network[1]. The cell-assembly hypothesis shifts the focus from the single cell to the

complete network activity. This view has led several laboratories to develop technology for simultaneous multi-cellular recording from a small region in the cortex[2, 3]. There remains, however, a large discrepancy between our ability to construct neural-network models and their correspondence with such multi-cellular recordings. To some extent this is due to the difficulty in observing simultaneous activity of any significant number of individual cells in a living nerve tissue. Extracellular electrophysiological measurements have so far obtained simultaneous recordings from just a few randomly selected cells (about 10), a negligibly small number compared to the size of the hypothesized cell-assembly. It is quite remarkable therefore, that such local measurements in the associative cortex have yielded so much information, such as synfire chains [2], multi-cell firing correlation[6], and statistical correlation between cell activity and external behavior. However, such observations have so far relied mostly on the accumulated statistics of cell firing over a large number of repeated experiments, to obtain any statistically significant effect. This is due to the very low firing rates (about 10Hz) of individual cells in the associative cortex, as can be seen in figure 1.

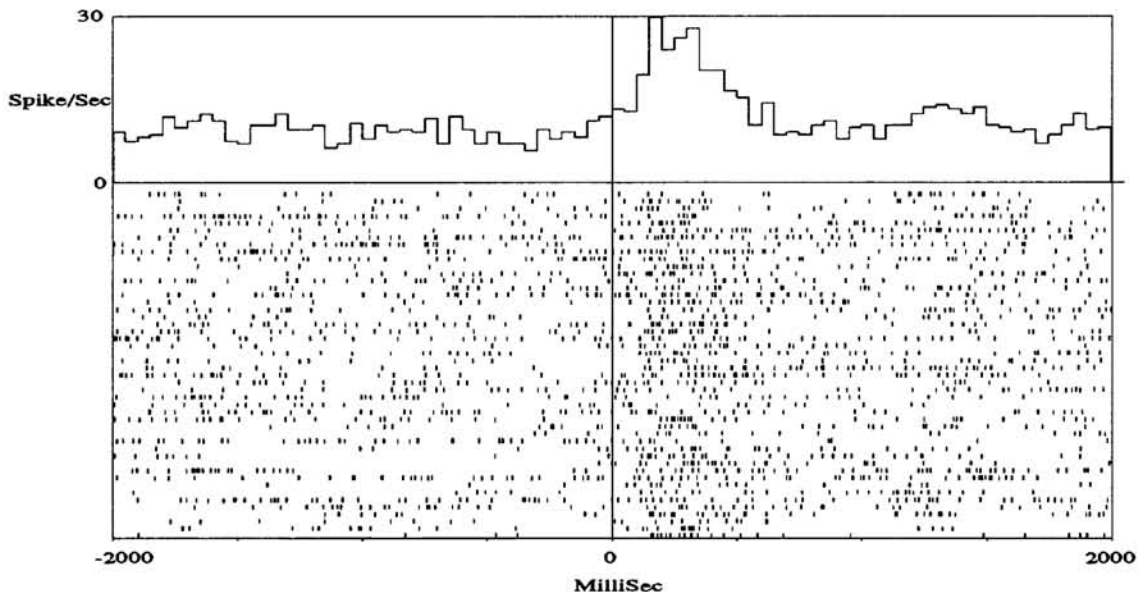

Figure 1: An example of firing times of a single unit. Shown are 48 repetitions of the same trial, aligned by the external stimulus marker, and drawn horizontally one on top of another. The accumulated histogram estimates the firing rate in 50msec bins, and exhibits a clear increase of activity right after the stimulus.

Clearly, simultaneous measurements of the activity of 10 units contain more information than single unit firing and pairwise correlations. The goal of the present study is to develop and evaluate a statistical method which can better capture the multi- unit nature of this data, by treating it as a vector stochastic process. The firing train of each of these units is conventionally modeled as a Poisson process with a time-dependent average firing rate[2]. Estimating the firing rate parameter requires careful averaging over a sliding window. The size of this window should be long enough to include several spikes, and short enough to capture the variability.

Within such a window the process is characterized by a vector of average rates, and possibly higher order correlations between the units.

The next step, in this framework, is to collect such vector-frames into statistically similar clusters, which should correspond to similar network activity, as reflected by the firing of these units. Furthermore, we can facilitate the well-established formulation of Hidden-Markov-Models[7] to estimate these "hidden" states of the network activity, similarly to the application of such models to other stochastic data, e.g. speech. The main advantage of this approach is its ability to characterize statistically the multi-unit process, in an unsupervised manner, thus allowing for finer discrimination of individual events. In this report we focus on the statistical discrimination of two behavioral modes, and demonstrate not only their distinct multi-unit firing patterns, but also the fact that the *coherency level* of the firing activity in these two modes is significantly different.

## 2   Origin of the data

The data used for the present analysis was collected at the Hadassa Medical School, by recording from a Rhesus monkey *Macaca Mulatta* who was trained to perform a spatial delayed release task. In this task the monkey had to remember the location from which a stimulus was given and after a delay of 1-32 seconds, respond by touching that location. Correct responses were reinforced by a drop of juice. After completion of the training period, the monkey was anesthetized and prepared for recording of electrical activity in the frontal cortex. After the monkey recovered from the surgery the activity of the cortex was recorded, while the monkey was performing the previously learned routine. Thus the recording does not reflect the learning process, but rather the cortical activity of the well trained monkey while performing its task. During each of the recording sessions six microelectrodes were used simultaneously. With the aid of two pattern detectors and four window-inscriminates, the activity of up to 11 single units (neurons) was concomitantly recorded. The recorded data contains the firing times of these units, the behavioral events of the monkey, and the electro-occulogram (EOG)[5, 2, 4].

### 2.1   Behavioral modes

To understand the results reported here it is important to focus on the behavioral aspect of these experiments. The monkey was trained to perform a spatial delayed response task during which he had to alternate between two behavioral modes. The monkey initiated the trial, by pressing a central key, and a fixation light was turned on in front of it. Then after 3-6 seconds a visual stimulus was given either from the left or from the right. The stimulus was presented for 100 millisec. After a delay the fixation light was dimmed and the monkey had to touch the key from which the visual stimulus came ("Go" mode), or keep his hand on the central key regardless of the external stimulus ("No-Go" mode). For the correct behavior the monkey was rewarded with a drop of juice. After 4 correct trials all the lights in front of the monkey blinked (this is called "switch" henceforth), signaling the monkey to change the behavioral mode - so that if started in the "Go" mode he now had to switch to "No-Go" mode, or vice versa.

There is a clear statistical indication, based on the accumulated firing histograms, that the firing patterns are different in these two modes. One of our main experimental results so far is a more quantitative analysis of this observation, both in terms of the firing patterns directly, and by using a new measure of the *coherency level* of the firing activity.

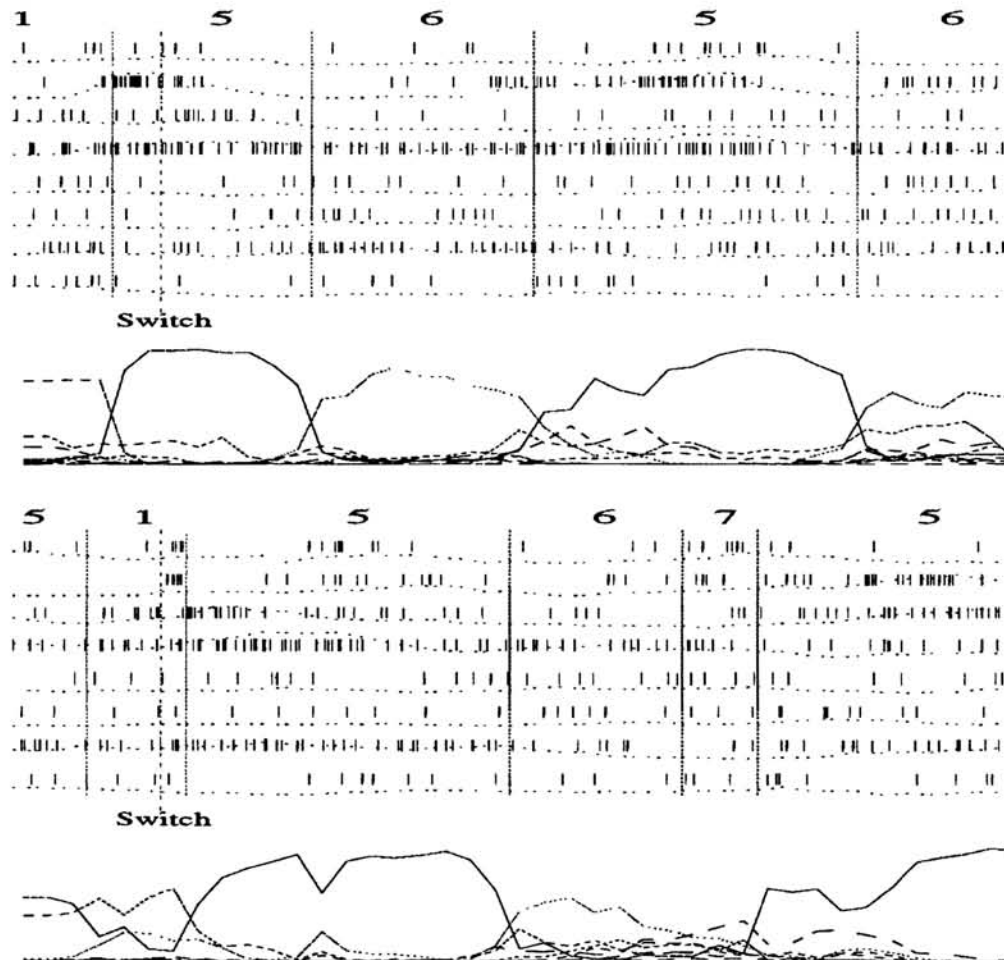

Figure 2: Multi-unit firing trains and their statistical segmentation by the model. Shown are 4 sec. of activity, in two trials, near the "switch". Estimated firing rates for each channel are also plotted on top of the firing spikes. The upper example is taken from the training data, while the lower is outside of the training set. Shown are also the association probabilities for each of the 8 states of the model. The monkey's cell-assembly clearly undergoes the state sequence "1", "5", "6", "5" in both cases. Similar sequence was observed near the same marker in many (but not all) other instances of the same event during that measurement day.

## 2.2   Method of analysis

As was indicated before, most of the statistical analysis so far was done by accumulating the firing patterns from many trials, aligned by external markers. This supervised mode of analysis can be understood from figure 1, where 48 different

"Go" firing trains of a single unit are aligned by the marker. There is a clear increase in the accumulated firing rate following the marker, indicating a response of this unit to the stimulus. In contrast, we would like to obtain, in an unsupervised *self organizing* manner, a statistical characterization of the *multi-unit* firing activity around the marked stimuli, as well as in other *unobserved* cortical processes. We claim to achieve this goal through characteristic sequences of Markov states.

## 3  Multivariate Poisson Hidden Markov Model

The following statistical assumptions underlie our model. Channel firing is distributed according to a Poisson distribution. The distances between spikes are distributed exponentially and their number in each frame, $n$, depends only on the mean firing rate $\lambda$, through the distribution

$$P_\lambda(n) = \frac{e^{-\lambda}\lambda^n}{n!} . \tag{1}$$

The estimation of the parameter $\lambda$ is performed in each channel, within a sliding window of $500ms$ length, every $100ms$. These overlapping windows introduce correlations between the frames, but generate less noisy, smoother, firing curves. These curves are depicted on top of the spike trains for each unit in figure 2.

The multivariate Poisson process is taken as a Maximum Entropy distribution with i.i.d. Poisson prior, subject to pairwise channel correlations as additional constraints, yielding the following parametric distribution

$$P_\Lambda(n_1, n_2, ..., n_d) = \prod_{i=1}^{d} P_{\lambda,}(n_i) \ \exp\left[ -\sum_{ij} \lambda_{ij}(n_i - \lambda_i)(n_j - \lambda_j) - \lambda_0 \right] . \tag{2}$$

The $\lambda_{ij}$ are additional Lagrange multipliers, determined by the observed pairwise correlation $E[(n_i - \lambda_i)(n_j - \lambda_j)]$, while $\lambda_0$ ensures the normalization. In the analysis reported here the pairwise correlation term has not been implemented.

The statistical distance between a frame and the cluster centers is determined by the probability that this frame is generated by the centroid distribution. This probability is asymptotically fixed by the empirical information divergence (KL distance) between the processes[8, 9]. For 1-dimensional Poisson distributions the divergence is simply given by

$$D[p_1|p_2] = \sum_x p_1(x) \log \frac{p_1(x)}{p_2(x)} = \lambda_2 - \lambda_1 + \lambda_1 \log \frac{\lambda_1}{\lambda_2} . \tag{3}$$

The uncorrelated multi-unit divergence is simply the sum of divergences for all the units. Using this measure, we can train a multivariate Poisson Hidden Markov Model, where each state is characterized by such a vector Poisson process. This is a special case of a method called *distributional clustering*, recently developed in a more general setup[10].

The clustering provides us with the desired statistical segmentation of the data into states. The probability of a frame, $x_t$, to belong to a given state, $S_j$, is determined by the probability that the vector firing pattern is generated by the state centroid's

distribution. Under our model assumptions this probability is a function solely of the empirical divergences, Eq.(3), and is given by

$$P(x_t \in S_j) = \frac{e^{-\beta D[x_t|S_j]}}{\sum_i e^{-\beta D[x_t|S_i]}} , \qquad (4)$$

where $\beta$ determines the "cluster-hardness". These state probability curves are plotted in figure 2 in correspondence with the spike trains. The most probable state at each instance determines the most likely segmentation of the data, and the frames are labeled by this most probable state number. These labels are also shown on top of the spike trains in figure 2.

## 4    Experimental results

We used about 6000 seconds of recordings done during a single day. It is important to note that this was an exceptionally good day in terms of the measurement quality. During that period the monkey performed 60 repetitions of his trained routine, in sets of 4 trials of "Go" mode, followed by 4 trials in the "No-Go" mode. We selected the 8 most active recorded units for our modeling. The training of the models was done on the first 4000 seconds of recording, 2000 seconds for each mode, while the rest was used for testing.

### 4.1    The nature of the segmentation

Any method can segment the data in some way, but the point is to obtain reliable predictions using this segmentation. As always, there is some arbitrariness in the choice of the number of states (or clusters), which ideally is determined by the data. Here we tested only 8 and 15 states, and in most cases 8 were sufficient for our purposes. Since we used "fuzzy", or "soft" clustering, each frame has some probability of belonging to any of the clusters. Although in most cases the most likely state is clearly defined, the complete picture is seen only from the complete association distribution. Notice, e.g., in the lower segment of figure 2, where a most likely state "7" "pops up" between states "6" and "5", but is clearly not significant, as seen from the corresponding probability curve.

### 4.2    Characterization of events by state-sequences

The first test of the segmentation is whether it is correlated with the external markers in any way. Since the markers were not used in any way during the training of the model (clustering), such correlations is a valid test of consistency. Moreover, one would like this correspondence to the markers to hold also outside of the training data. An exhaustive statistical examination of this question has not been made, as yet, but we could easily find many instances of similar state sequences near the same external marker, both within and outside of the training data. In figure 2 we bring a typical example to this effect. The next step is to train small left-to-right Markov models to spot these events more reliably.

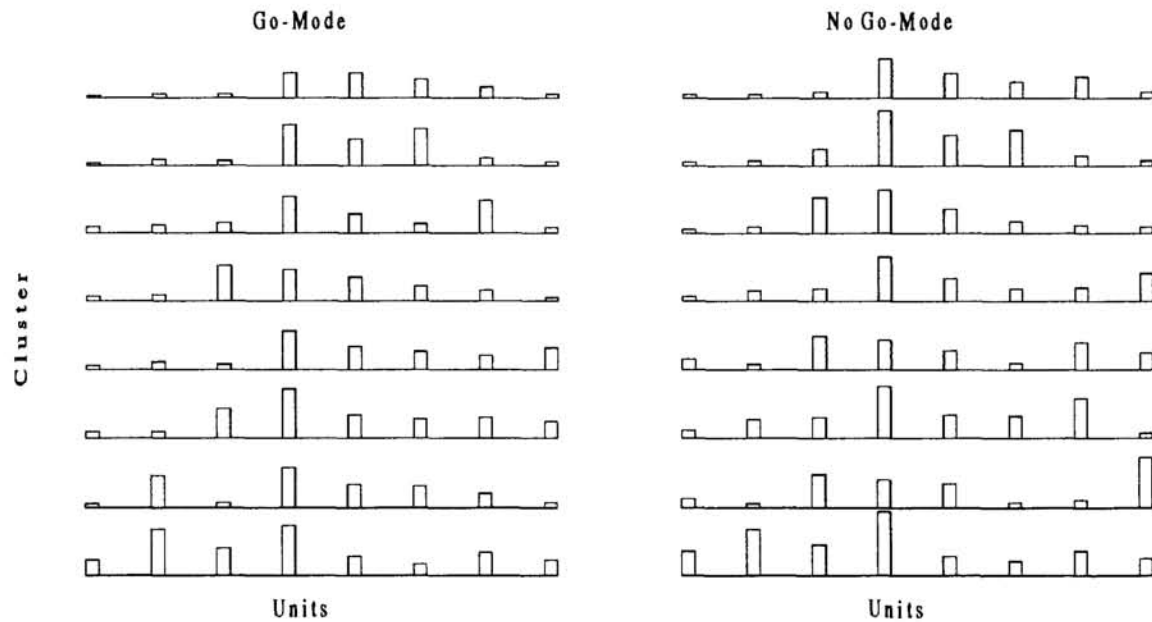

Figure 3: Average firing rates for each unit in each state, for the "Go" and "No-Go" modes. Notice that while no single unit clearly discriminates the two modes, their overall statistical discrimination is big enough that on average 100 frames are enough to determine the correct mode, more than 95% of the time.

## 4.3   Statistical Inference of "Go" and "No-Go" modes

Next we examined the statistical difference between models trained on the "Go" vs. "No-Go" modes. Here we obtained a highly significant difference in the cluster centroid's distributions, as shown in figure 3. The average statistical divergence between different clusters within each mode were 9.18 and 9.52 (natural logarithm),in "Go" and "No-Go" respectively, while in between those modes the divergence was more than 35.

## 4.4   Behavioral mode and the network firing coherency

In addition to the clearly different cluster centers in the two modes, there is another interesting and unexpected difference. We would like to call this *firing coherency level*, and it characterize the spread of the data around the cluster centers. The average divergence between the frames and their most likely state is consistently much higher in the "No-Go" mode than in the "Go" mode (figure 4). This is in agreement with the assumption that correct performance of the "No-Go" paradigm requires little attention, and therefore the brain may engage in a variety of processes.

## Acknowledgments

Special thanks are due to Moshe Abeles for his continuous encouragement and support, and for his important comments on the manuscript. We would also like to thank Hagai Bergman, and Eilon Vaadia for sharing their data with us, and for numerous stimulating and encouraging discussions of our approach. This research was supported in part by a grant from the Unites States Israeli Binational Science Foundation (BSF).

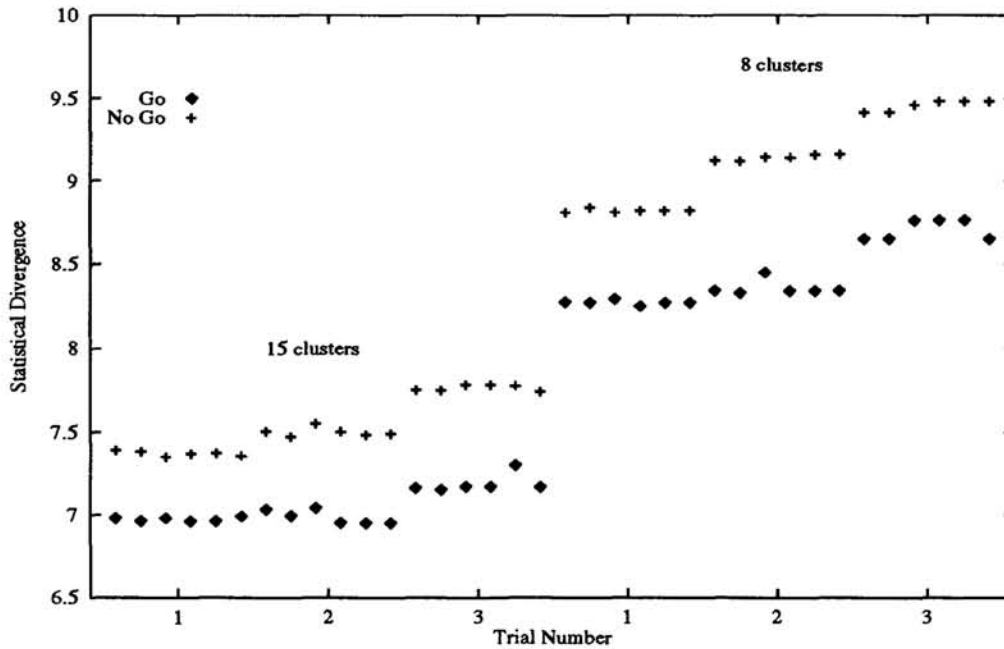

Figure 4: Firing coherency in the two behavioral modes at different clustering trials. The "No-Go" average divergence to the cluster centers is systematically higher than in the "Go" mode. The effect is shown for both 8 and 15 states, and is even more profound with 8 states.

## Footnotes

*{itay,tishby}@cs.huji.ac.il

# References

[1] D. O. Hebb, *The Organization of Behavior*, Wiley, New York (1949)

[2] M. Abeles, *Corticonics*, (Cambridge University Press, 1991)

[3] J. Kruger, *Simultaneous Individual Recordings From Many Cerebral Neurons: Techniques and Results*, Rev. Phys. Biochem. Pharmacol.: 98:pp. 177-233 (1983)

[4] M. Abeles, E. Vaadia, H. Bergman, *Firing patterns of single unit in the prefrontal cortex and neural-networks models.*, Network 1 (1990)

[5] M. Abeles, H. Bergman, E. Margalit and E. Vaadia, *Spatio Temporal Firing Patterns in the Frontal Cortex of Behaving Monkeys.*, Hebrew University preprint (1992)

[6] E. Vaadia, E. Ahissar, H. Bergman, and Y. Lavner, *Correlated activity of neurons: a neural code for higher brain functions* in: J.Kruger (ed), Neural Cooperativity pp. 249-279, (Springer-Verlag 1991).

[7] A. B. Poritz, *Hidden Markov Models: A Guided tour*,(ICASSP 1988 New York).

[8] T.M. Cover and J.A. Thomas, *Information Theory*, (Wiley, 1991).

[9] J. Ziv and N. Merhav, *A Measure of Relative Entropy between Individual Sequences*, Technion preprint (1992)

[10] N. Tishby and F. Pereira, *Distributional Clustering*, Hebrew University preprint (1993).
